# Decoding Ipsilateral Finger Movements from ECoG Signals in Humans

Yuzong Liu[1], Mohit Sharma[2], Charles M. Gaona[2], Jonathan D. Breshears[3], Jarod Roland [3],
Zachary V. Freudenburg[1], Kilian Q. Weinberger[1], and Eric C. Leuthardt[2,3]

[1]Department of Computer Science and Engineering, Washington University in St. Louis
[2]Department of Biomedical Engineering, Washington University in St. Louis
[3]Department of Neurosurgery, Washington University School of Medicine

## Abstract

Several motor related Brain Computer Interfaces (BCIs) have been developed over the years that use activity decoded from the contralateral hemisphere to operate devices. Contralateral primary motor cortex is also the region most severely affected by hemispheric stroke. Recent studies have identified ipsilateral cortical activity in planning of motor movements and its potential implications for a stroke relevant BCI. The most fundamental functional loss after a hemispheric stroke is the loss of fine motor control of the hand. Thus, whether ipsilateral cortex encodes finger movements is critical to the potential feasibility of BCI approaches in the future. This study uses ipsilateral cortical signals from humans (using ECoG) to decode finger movements. We demonstrate, for the first time, successful finger movement detection using machine learning algorithms. Our results show high decoding accuracies in all cases which are always above chance. We also show that significant accuracies can be achieved with the use of only a fraction of all the features recorded and that these core features are consistent with previous physiological findings. The results of this study have substantial implications for advancing neuroprosthetic approaches to stroke populations not currently amenable to existing BCI techniques.

## 1   Introduction

**Note by authors after publication: The results in Figure 3 could not be reproduced in subsequent experiments and should be considered invalid. We apologize for this mishap. Other results in this paper are not affected.**   The evolving understanding of motor function in the brain has led to novel Brain Computer Interface (BCI) platforms that can potentially assist patients with severe motor disabilities. A BCI is a device that can decode human intent from brain activity alone in order to create an alternate communication and control channel for people with severe motor impairments [39]. This brain-derived control is dependent on the emerging understanding of cortical physiology as it pertains to motor function. Examples are seen in the seminal discoveries by Georgopoulus and Schwartz that neurons in motor cortex show directional tuning and, when taken as a population, can predict direction and speed of arm movements in monkey models [12, 19]. In the subsequent two decades, these findings were translated to substantial levels of brain-derived control in monkey models and preliminary human clinical trials [14, 34]. Another example is seen in Pfurtschellers work in analyzing electroencephalography (EEG). His group was one of the first to describe the changes in amplitudes in sensorimotor rhythms associated with motor movement [24]. As a result, both Pfurtscheller and Wolpaw have used these signals to achieve basic levels of control in humans with amyotrophic lateral sclerosis (ALS) and spinal cord injury [25, 40]. All these methods are based on a functioning motor cortex capable of controlling the contralateral limb. This is the

exact situation that does not exist in unilateral stroke. Hence, these systems to date offer little hope for patients suffering from hemispheric stroke. For a BCI to assist a hemiparetic patient, the implant will likely need to utilize unaffected cortex ipsilateral to the affected limb (opposite the side of the stroke). To do so, an expanded understanding of how and to what degree of complexity motor and motor associated cortex encodes ipsilateral hand movements is essential.

Electrocorticography (ECoG), or signal recorded from the surface of the brain, offers an excellent opportunity to further define what level of motor information can be deciphered from human ipsilateral cortex related to movements (e.g. gross motor movements versus fine motor kinematics of individual finger movements). The ECoG signal is more robust compared to the EEG signal: its magnitude is typically five times larger, its spatial resolution as it relates to independent signals is much greater (0.125 versus 3.0 cm for EEG), and its frequency bandwidth is significantly higher (0-550 Hz versus 0- 40 Hz for EEG) [11, 30]. When analyzed on a functional level, many studies have revealed that different frequency bandwidths carry highly specific and anatomically focal information about cortical processing. Thus far, however, no studies have utilized these ECoG spectral features to definitively analyze and decode cortical processing of the specific kinematics of ipsilateral finger movements.

In the past year, the first demonstration of this concept of utilizing ipsilateral motor signals for simple device control have been published both with ECoG (in healthy subjects) and MEG (in stroke patients) [4, 38]. In this study we set out to further explore the decoding of individual finger movements of the ipsilateral hand that could potentially be utilized for more sophisticated BCIs in the future. We studied 3 subjects who required invasive monitoring for seizure localization. Each had electrode arrays placed over the frontal lobe and a portion of sensorimotor cortex for approximately a week. Each subject performed individual finger tasks and the concurrent ECoG signal was recorded and analyzed. The principal results show that individual ipsilateral finger movements can be decoded with high accuracy. Through machine learning techniques, our group was able to determine the intent to flex and extend individual finger movements of the ipsilateral hand. These results indicate that an ECoG based BCI platform could potentially operate a hand orthotic based on ipsilateral motor signals. This could provide a neuroprosthetic alternative to patients with hemispheric stroke who have otherwise failed non-invasive and medical rehabilitative techniques.

## 2 Data Collection

The subjects in this study were three patients (females; 8, 36, 48 years of age) with intractable epilepsy who underwent temporary placement of intracranial electrode arrays to localize seizure foci prior to surgical resection. All had normal levels of cognitive function and all were right-handed. Subject 1 had a right hemispheric $8 \times 8$ grid while subjects 2 and 3 had left hemispheric $8 \times 8$ grids. All gave informed consent. The study was approved by the Washington University Human Research Protection Office.

Each subject sat in their hospital bed 75 cm from a 17-inch LCD video screen. In this study, the subject wore a data glove on the each hand to precisely monitor finger movements. Each hand rested on a table in front of the screen. The screen randomly cued the patient to flex and extend a given finger (e.g., left index finger, right ring finger, etc.). A cue came up on the monitor and as long as it was present, subjects would, at a self-paced speed, move the indicated finger from the flexed to the extended position until the cue disappeared. They were instructed on the method prior to participation. Each cued task period would last 2 seconds with a randomized rest period between 1.5 and 2.5 seconds(i.e., a trial). There were on average 30 trials per finger for a given subject. For subject 1, the thumb data recording was found to be noisy and hence was eliminated from any further analysis. Visual cues were presented using the BCI2000 program [27]. All motor hand kinematics were monitored by the patient wearing a USB linked 5DT Data Glove 5 Ultras (Fifth Dimension, Irvine, CA) on each hand. These data gloves are designed to measure finger flexure with one sensor per finger at up to 8-bit flexure resolution. The implanted platinum electrode arrays were $8 \times 8$ electrode arrays(Ad-Tech, Racine, WI and PMT, Chanhassen, MN). The grid and system setup details are described elsewhere [38]. ECoG signals were acquired using BCI2000, stored, and converted to MATLAB files for further processing and analysis. All electrodes were referenced to an inactive intracranial electrode. The sampling frequency was 1200 Hz and data acquisition is band-pass filtered from 0.15 to 500 Hz.

## 2.1 Data Preprocessing

**Gabor Filter Analysis** All ECoG data sets were visually inspected and re-referenced with respect to the common average to account for any unwanted environmental noise. For these analyses, the time-series ECoG data was converted into the frequency domain using a Gabor filter bank [17]. Spectral amplitudes between 0 and 550 Hz were analyzed on a logarithmic scale. The finger positions from the data glove were converted into velocities. These frequency responses and velocities were then used as an input to machine learning algorithms described below. Inherent in this is the estimation of the lag between the ECoG signal and the actual finger movement. As part of the modeling process, the value of this variable which resulted in the best decoding accuracy was chosen for further analysis. Average time lags were then used to align the ECoG signal to the finger movement signal. Those features optimized for predicting individual finger movement were then reviewed in light of anatomic location and spectral association in each subject.

**Dimensionality Reduction** Due to the high dimensionality of the spectral data ($\#channels(N) \times \#frequencies(F)$), it is important to reduce the dimensions in order to build a more conducive machine learning algorithm. Principle component analysis, or PCA, is among the most popular dimensionality reduction algorithm. PCA projects the original high-dimensional feature space into a much lower *principle subspace*, such that the variance of low-dimensional data is maximized. In the real-time decoding task, we use PCA to reduce the input data. However, in the weight analysis, we preserve all the $N \times F$ features because we want to study the effect of using all the features.

**Electrode Co-Registration** Radiographs were used to identify the stereotactic coordinates of each grid electrode [10], and cortical areas were defined the GetLOC package for ECoG electrode localization [18]. Stereotactically defined electrodes were mapped to the standardized brain model. The experimental results were then collated with these anatomical mapping data.

## 3 Algorithms

In this section, we describe the machine learning algorithms used for the finger movement decoding tasks. We focus on three different settings: 1. binary classification, 2. multiclass classification and 3. multitask classification. All the data is split into a training and a testing dataset. We chose our parameters based on a validation dataset split from the training dataset.

**Binary Classification** We treat the finger movement detection problem as a binary classification setting. The data is presented as a time series with feature vector $\mathbf{x}_t$ and velocity label $y_t$ at time $t$. The goal is to predict if at time $t$, a finger is moving ($y_t = 1$) or not ($y_t = -1$).

For this purpose, we adapted logistic regression (LR) [26] and binary support vector machines (SVM) [7]. Both classifiers learn parameters $(\mathbf{w}, b) \in \mathcal{R}^d \times \mathcal{R}$. The prediction at time $t$ is computed as $\hat{y}_t = \text{sign}(\mathbf{w}^\top \mathbf{x}_t + b)$. The vector $\mathbf{w}$ is learned with the following optimization problem

$$\min_{(\mathbf{w}, b)} \sum_{t=1}^{T} \mathcal{L}(\mathbf{w}^\top \mathbf{x}_t + b, y_t) + \lambda |\mathbf{w}|_q. \tag{1}$$

Here, $\lambda \geq 0$ is the regularization constant that trades off weight sparsity with complexity. The norm of the regularization can be the $\ell 1$ norm ($q = 1$) or the $\ell 2$ norm ($q = 2$). The $\ell 1$ norm has the tendency to result in sparse classifiers which assign non-zero weights to only a small subset of the available features. This allows us to infer which brain regions and frequencies are most important for accurate predictions. The $\ell 2$ norm tends to yield slightly better classification results (and is easier to optimize) but is not as interpretable as it typically assigns small weights to many features. The loss functions $\mathcal{L}$ differ for the two above mentioned algorithms. We will denote the loss function for logistic regression as $\mathcal{L}_{lr}$ and for SVMs as $\mathcal{L}_{svm}$. The exact definitions are:

$$\mathcal{L}_{lr}(\mathbf{z}, y) = \log(1 + \exp(-y\mathbf{z})) \qquad \mathcal{L}_{svm}(\mathbf{z}, y) = \max(1 - y\mathbf{z}, 0) \tag{2}$$

**Multiclass Classification** A second setting of interest is the differentiation of fingers. Here we do not predict *if* a finger is moving but *which one*. Consequently, at any time point $t$ we could have one of $K$ possible labels, such as "Index Finger" ($y_t = 1$), "Ring Finger" ($y_t = 2$), etc. We adopt the Crammer and Singer multi-class adaptation of support vector machines (MCSVM) [8]. For each class $k \in \{1, \ldots, K\}$, we learn class-specific parameters $\mathbf{w}_k, b_k$. The loss only focuses on

pairwise comparisons between the different classes and ensures that $\mathbf{w}_k^\top \mathbf{x}_t + b_k \geq \mathbf{w}_r^\top \mathbf{x}_t + b_r + 1$ if $y_t = k$ for any $r \neq k$. For completeness, we re-state the optimization problem:

$$\min_{(\mathbf{w}_1,b_1),...,(\mathbf{w}_K,b_K)} \sum_{t=1}^{T} \sum_{r \neq y_t} \max(1 + \mathbf{w}_r^T \mathbf{x}_t + b_r - (\mathbf{w}_{y_t}^T \mathbf{x}_{y_t} + b_{y_t}), 0) + \lambda \sum_{k=1}^{K} |\mathbf{w}_k|_q. \quad (3)$$

Similar to the scenario of binary classification, the constant $\lambda \geq 0$ regulates the trade-off between complexity and sparseness.

**Multitask Learning** In the movement detection setting, each finger is learned as an independent classification problem. In the finger discrimination setting, we actively discriminate between the individual fingers. Multitask learning (MTL) is a way to combine the binary finger movement detection problems by learning them jointly [5]. In the setting of brain decoding, it seems reasonable to assume that there are certain features which are associated with the general cortical processing of finger movements. This is analogous to the notion of language processing and articulation in cortical areas. Functional magnetic resonance imaging (fMRI) studies have shown that although speech is represented in general cortical areas, individual features specific to different kinds of words can be found [16, 23]. We adopt the MTL adaptation for SVMs of [9], and an analogous framework for logistic regression, which leverages the commonalities across learning tasks by modeling them explicitly with an additional shared weight vector $\mathbf{w}_0$. The prediction at time $t$ for finger $k$ is defined as $\hat{y}_t = (\mathbf{w}_0 + \mathbf{w}_k)^\top \mathbf{x}_t$. The corresponding optimization problem becomes

$$\min_{\mathbf{w}_0, \mathbf{w}_1,...,\mathbf{w}_K} \lambda_0 |\mathbf{w}_0| + \sum_{k=0}^{K} \sum_{t=1}^{T} \mathcal{L}((\mathbf{w}_0 + \mathbf{w}_k)^\top \mathbf{x}_t, y_t) + \lambda_k |\mathbf{w}_k|_q. \quad (4)$$

The parameter $\lambda_0$ regulates how much of the learning is shared. If $\lambda_0 \to +\infty$, then $\mathbf{w}_0 = \mathbf{0}$ and we reduce our setting to the original binary classification mentioned above. On the other hand, setting $\lambda_0 = 0$ and $\lambda_{k>0} \gg 0$ will result in weight vectors $\mathbf{w}_{k>0} = \mathbf{0}$. As a result, one would learn only a single classifier with weight vector $\mathbf{w}_0$ for generic finger movement.

## 4   Results

In this section we evaluate our algorithms for ipsilateral decoding on three subjects. First, we approximate the time-lag between ECoG signal and finger movement, then we present decoding results on finger movement detection, discrimination and also joint decoding of all fingers in one hand.

**Time Lag** We first study the effects of decoding time lag between cortical signal and movement using features. The decoding accuracy is computed by shifting the feature dataset $x_t$ and the target dataset $y_t$ by a presumed number of sample points (i.e. we are evaluating the performance of decoder h: $h(x_t) = y_{t+\delta_T}$, by increasing the value of $\delta_T$). The best time lag is selected as the value of $\delta_T$ which leads to best decoding accuracy. Figure 1 shows the decoding accuracy as a function of time-lag for four individual finger movements in Subject 1. Offsets between 0 and 800 ms are tested for all fingers and an average offset time is computed. The average time lag for the ipsilateral finger movement for Subject 1 is observed to be around 158 ms. This is in accordance with previous studies by our group which show similar time lags between cortical activity

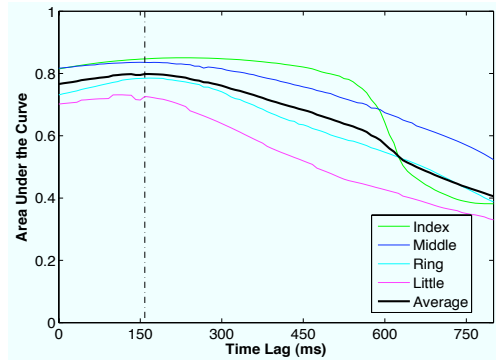

Figure 1: Decoding time lag for ipsilateral finger movement in Subject 1. The x-axis is the presumed time lag $\delta_T$ (ms) between input feature vectors and target labels, and the y-axis is the area under the ROC curve computed from $\mathcal{L}1$-regularized logistic regression model. The **bold** black line is the average AUC, and the best decoding time-lag is indicated by the black dotted line.

and actual movements [38]. All further analysis is based on cortical activity (features) shifted relative to movement by the average time-lag reported here.

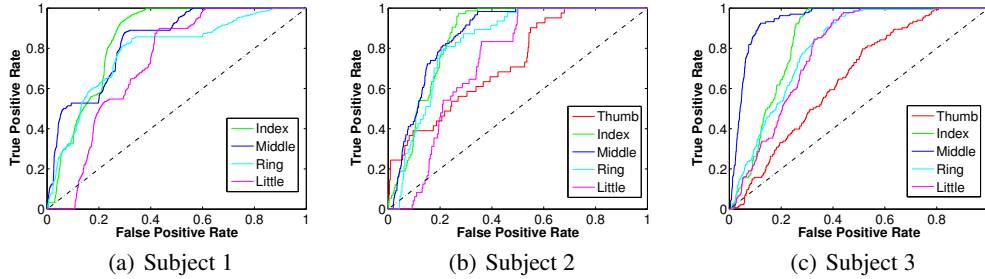

(a) Subject 1         (b) Subject 2         (c) Subject 3

Figure 2: ROC curve for the ipsilateral finger movement decoder. Horizontal axis shows the false positive rate, and the vertical axis shows the true positive rate. The dotted line is the accuracy of a random classifier. Classifiers that have higher *area under the ROC curve*, or AUC, indicate better classification performance.

**Detecting Finger Movement** We characterize the movement detection task as a binary classification. We first set a threshold $thresh$, and label the targets $y_t$ as 1 if the velocity at time t $v_t \geq thresh$, and -1 otherwise. Then, we use $\ell$1-regularized logistic regression for the binary classification. We use receiver operating characteristic (ROC) curve to evaluate the performance of the binary classification. ROC curve is widely used in signal estimation and detection theory, and is a graphical plot of true positive rate versus the false positive rate. ROC analysis allows user to pick the optimal discrimination threshold for the binary classifier. We pick regularizer $\lambda$ from validation dataset. Figure 2 shows the result of ROC curve for three subjects. This demonstrates that $\ell$1-regularized logistic regression is a powerful tool in detecting finger movement.

**Finger Discrimination** In this section, we study how to discriminate *which* finger has made the movement. We first extract the sample points of which the finger is moving from the time-series. We then apply multiclass SVM to do the classification. The result is shown as the confusion matrices in Figure 3, and the colorbar shows the accuracy. Each row of the matrix represents the finger that actually moved and each column represents predicted finger. The elements of the matrix shows the percentage of all movements of a particular finger that has been classified as particular predicted finger. Note that the accuracy by a random multiclass classifier is 1/(number of fingers). It can be concluded that the ECoG signal contains useful information to discriminate individual finger movement.

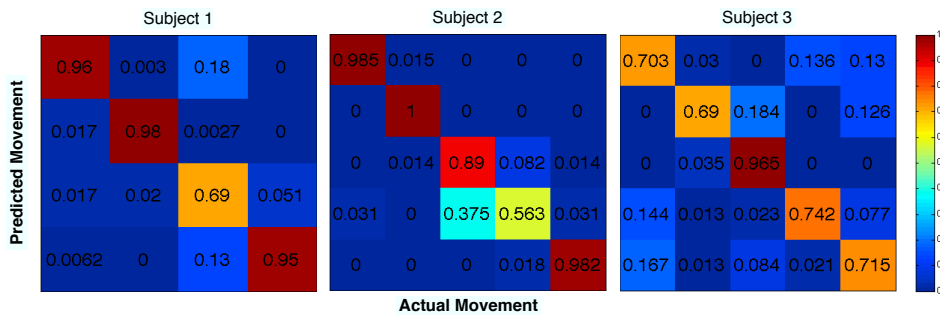

Figure 3: **Note from authors after publication: Results in this figure are invalid (see note in introduction).**Confusion matrix of finger movement multiclass classification. The rows are the actual movement, and the columns are the predicted movement.

## 4.1 Learning Commonality from the Brain Activity

In this section, we present how multitask learning improves the performance of the classifier. Although multitask learning has been employed in the context of brain signal decoding [2], we are the first to decode ECoG signals in humans. We group all the individual finger movement together, such that each task has similarity with others. First of all, we evaluate the performance of single-task

learning using SVM. Then, we study the SVM-based multitask learning. As we show in Equation 4, we make trade-off between modeling joint component and and modeling class-specific components by adjusting parameters $\lambda_0$ and $\lambda$. We search a number of regularization constant $(\lambda_0, \lambda)$, and pick up the parameters that lead to highest average AUC for all tasks. Table 1 shows the comparison of SVM-based single task learning and multitask learning. Here we evaluate the multitask learning algorithm based on the improvement of (1-AUC); (1-AUC) stands for the area above the curve. The average improvement of the decoder for three patients is 25.53%, 5.60%, and 18.57%, respectively. This confirms our assumption that there exists brain activity that controls the finger movement, irrespective of any particular finger. By carefully searching the best parameters that regulates the trade-off between learning commonality among all finger movement and specificity of exact finger movement, the classification algorithm can be significantly improved. We also compare the $\ell 1/\ell 2$-regularized logistic regression-based multitask learning with SVM-based multitask learning. There is an improvement on (1-AUC) for logistic regression-based multitask learning. Again, it illustrates that multitask learning is particularly helpful in learning similar tasks that are controlled by the brain. However, we prefer SVM-based multitask learning because of the larger improvement.

| AUC | Subject 1 | | Subject 2 | | Subject 3 | |
|---|---|---|---|---|---|---|
| | STL | MTL | STL | MTL | STL | MTL |
| Thumb | N/A | N/A | 0.7710 | **0.7845** | 0.7680 | **0.8611** |
| Index | 0.8477 | **0.8494** | **0.9061** | 0.8948 | 0.7454 | **0.8242** |
| Middle | 0.8393 | **0.8569** | **0.9021** | 0.8990 | 0.9459 | **0.9481** |
| Ring | 0.8000 | **0.8561** | 0.8888 | **0.8894** | 0.7404 | **0.7479** |
| Little | 0.7425 | **0.7865** | 0.7124 | **0.7586** | 0.7705 | **0.7801** |

Table 1: Comparison of SVM-based single-task learning (STL) and SVM-based multi-task learning (MTL). The parameters are chosen from validation dataset: $\lambda_0 = 10^{-2}$ and $\lambda = 10^4$ for Subject 1, $\lambda_0 = 1$ and $\lambda = 10^2$ for Subject 2, and $\lambda_0 = 10^2$ and $\lambda = 10^{-2}$ for Subject 3. The best decoding performance is indicated in **bold**.

## 5   Weight Analysis

An important part of decoding finger movements from cortical activity is to map the features back to cortical domain. Physiologically, it is important to understand the features which contribute most to the decoding algorithms i.e. the features with the highest weights. As shown in Table 2 below, the decoding accuracy, indicated by AUC, does not change much as we increase the number of features used for classification. This signifies that from the large feature set used for decoding, a few features form the core and are the most important. To visualize these core features, we mapped the top 30 features back to the brain. Figure 4 above shows the normalized weights from the features used to classify finger movements from non-movements. It is apparent from the figure that the features with the highest weights fall in the DLPFC and premotor areas. This is what we would expect since these two areas are the one's most involved in the planning of motor movements. As previously reported, the frequency range with the highest weights falls in the lower frequencies in ipsilateral movements [38]. In our case, the frequencies fall in the delta-alpha range. As noted by Tallon-Baudry, attention networks of the brain affect the oscillatory synchrony as low as theta-alpha range frequencies [31].

| # features | 1 | 2 | 4 | 8 | 16 | 32 | 64 | 256 | 4096 |
|---|---|---|---|---|---|---|---|---|---|
| AUC | 0.681 | 0.717 | 0.755 | 0.787 | 0.803 | 0.807 | 0.807 | 0.807 | 0.808 |

Table 2: The area under the curve (AUC) as a function of the number of features used for classification. Features were selected in decreasing order of their respective absolute weights from logistic regression with $\ell 1$ regularization.

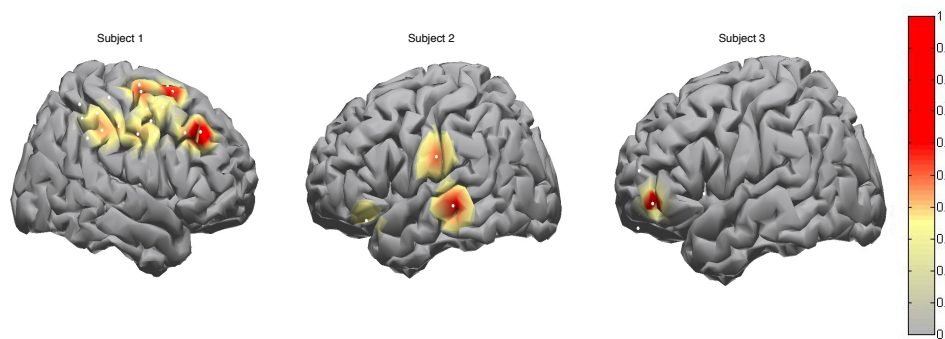

Figure 4: Brain map representing the weights of the top 30 features of the three subjects. It represents the variability in cortical processing of ipsilateral finger movements. It can also be seen that cortical processing occurs as a network involving dorsolateral prefrontal cortex, pre-motor and motor areas. The frequency range for these features is in the delta and alpha range i.e. the low frequency range.

## 6 Discussion

The notion that motor cortex plays a role in ipsilateral body movements was first asserted by Nyberg-Hansen et al. that 15% of corticospinal neurons did not decussate in cats [22]. Originally this was felt to represent more axial motor control. Further studies in single-neuron recordings in monkey models extended this observation to include ipsilateral hand and finger function. Tanji et al. demonstrated that a small percentage of primary motor cortical neurons showed increased activity with ipsilateral hand movements [32]. This site was found to be anatomically distinct from contralateral hand sites and, when stimulated, produced ipsilateral hand movements [1]. Additionally, a larger subset of premotor neurons was found to demonstrate more robust activations with cues to initiate movement during both ipsilateral and contralateral movements than with primary motor sites [3, 6]. These findings in animal models support the conclusion that a small percent of motor and a larger percent of premotor cortex participate in control of ipsilateral limb and hand movements.

In humans, there appears to be a dichotomy in how motor regions contribute depending on whether the primary or non-primary motor cortex is examined. Using fMRI Newton et al. demonstrated that there was a negative change from baseline in fMRI bold sequence in M1 associated with ipsilateral movements and postulated this to represent increased inhibition [21]. Verstynen et al., however, recently published contrasting results. Their group showed that anatomically distinct primary motor sites demonstrated increased activation that became more pronounced during the execution of complex movements [36]. The role that premotor cortex plays appears to be distinct from that of primary motor cortex. In normal subjects, fMRI shows that there is more robust bilateral activation of the dorsal premotor cortex with either contralateral or ipsilateral hand movements [15]. The findings by Huang, et al. (2004) demonstrated that ipsilateral premotor areas have magnetoencephalography (MEG) dipole peak latencies that significantly precede contralateral M1 sensorimotor cortex in performing unilateral finger movements. Using electroencephalography (EEG), ipsilateral hand movements have been shown to induce alteration in cortical potentials prior to movement; this is referred to as premotor positivity [33, 29]. Spectral analyses of EEG signals have shown bihemispheric low-frequency responses with various finger and hand movements. Utilizing electrocorticography (ECoG), Wisneski et al more definitively demonstrated that the cortical physiology associated with ipsilateral hand movements was associated with lower frequency spectral changes, an earlier timing, and premotor predominant cortical localization, when compared to cortical physiology that was associated with contralateral hand movements [38]. Taken together, these findings support more of a motor planning role, rather than execution role, in ipsilateral hand actions.

Decoding the information present in the ECoG signal with regard to ipsilateral finger movements is important in defining the potential use of BCI methodologies for patients with hemispheric dysfunction due to stroke or trauma. If high resolution motor kinematics can be decoded from the ECoG signal (e.g. individual finger flexion and extension), a BCI platform could potentially be created to restore function to a stroke induced paretic hand. Since up to one-half of hemispheric stroke

patients are chronically left with permanent loss of function in their affected hand, this could have substantial clinical impact [20]. Functional imaging has shown these severely affected patients to have increased activity in the premotor regions of their unaffected hemispheres [28, 37]. The exact role this activity plays is still unclear. It may simply be an indicator of a more severe outcome [35] or an adapative mechanism to optimize an already poor situation [13]. Thus, incomplete recovery and its association with heightened ipsilateral activation may reflect the up-regulation of motor planning with an inability to execute or actuate the selected motor choice. In this situation, a BCI may provide a unique opportunity to aid in actuating the nascent premotor commands. By decoding the brain signals associated with a given motor intention, the BCI may then convert these signals into commands that could control a robotic assist device that would allow for improved hand function (i.e., a robotic glove that opens and closes the hand or a functional electrical simulator that operates the nerves and muscles of the hand). The BCI would allow the ipsilateral premotor cortex to bypass the physiological bottleneck determined by injured and dysfunctional contralateral primary cortex (due to stroke) and the small and variable percentage of uncrossed motor fibers from ipsilateral M1. This new methodology would allow for restoration of function in chronically and severely affected subjects for whom methods of rehabilitation have not accomplished a sufficiently recovery.

# 7   Conclusion

To our knowledge, this work describes the first instance of successful detection of individual finger movements from human ipsilateral ECoG signals. In this paper, we present a general decoding framework using the following algorithms: (1) $\ell 1$-regularized logistic regression for detecting finger movement; (2) Multiclass support vector machines to discriminate between fingers; and (3) First demonstration of multitask learning into the ECoG signal to improve decoding accuracy. The results presented here suggest that there exists information on the cortex ipsilateral to the moving fingers which can be decoded with high accuracy using machine learning algorithms. These results present a great potential in the world of neuroprosthetics and BCI. For patients suffering from stroke and hemiparesis, decoding finger movements from the unaffected hemisphere can be of tremendous help. Our future goals involve simultaneous decoding of finger and arm movements (using standard center out joystick task) from both ipsilateral and contralateral hemispheres. Another important goal is the real-time use of these decoding results and demonstrate their utility in the world of BCI.

# References

[1] H. Aizawa, H. Mushiake, M. Inase, and J. Tanji. An output zone of the monkey primary motor cortex specialized for bilateral hand movement. *Experimental Brain Research*, 82(1):219–221, 1990.

[2] M. Alamgir, M. Grosse-Wentrup, and Y. Altun. Multitask learning for brain-computer interfaces. *Proceedings of the Thirteenth International Conference on Artificial Intelligence and Statistics*, 9:17–24, 2010.

[3] C. Brinkman and R. Porter. Supplementary motor area in the monkey: activity of neurons during performance of a learned motor task. *Journal of Neurophysiology*, 42(3):681, 1979.

[4] E. Buch, C. Weber, L. Cohen, C. Braun, M. Dimyan, T. Ard, J. Mellinger, A. Caria, S. Soekadar, A. Fourkas, et al. Think to move: a neuromagnetic brain-computer interface (BCI) system for chronic stroke. *Stroke*, 39(3):910, 2008.

[5] R. Caruana. Multitask learning. *Machine learning*, 28:41–75, 1997.

[6] P. Cisek, D. Crammond, and J. Kalaska. Neural activity in primary motor and dorsal premotor cortex in reaching tasks with the contralateral versus ipsilateral arm. *Journal of neurophysiology*, 89(2):922, 2003.

[7] C. Cortes and V. Vapnik. Support-vector networks. *Machine learning*, 20(3):273–297, 1995.

[8] K. Crammer and Y. Singer. On the algorithmic implementation of multiclass kernel-based vector machines. *The Journal of Machine Learning Research*, 2:265–292, 2002.

[9] T. Evgeniou and M. Pontil. Regularized multi–task learning. In *KDD*, pages 109–117, 2004.

[10] P. Fox, J. Perlmutter, and M. Raichle. A stereotactic method of anatomical localization for positron emission tomography. *Journal of Computer Assisted Tomography*, 9(1):141, 1985.

[11] W. Freeman, M. Holmes, B. Burke, and S. Vanhatalo. Spatial spectra of scalp eeg and emg from awake humans. *Clinical Neurophysiology*, 114(6):1053–1068, 2003.

[12] A. Georgopoulos, J. Kalaska, R. Caminiti, and J. Massey. On the relations between the direction of two-dimensional arm movements and cell discharge in primate motor cortex. *Journal of Neuroscience*, 2(11):1527, 1982.

[13] C. Gerloff, K. Bushara, A. Sailer, E. Wassermann, R. Chen, T. Matsuoka, D. Waldvogel, G. Wittenberg, K. Ishii, L. Cohen, et al. Multimodal imaging of brain reorganization in motor areas of the contralesional hemisphere of well recovered patients after capsular stroke. *Brain*, 129(3):791, 2006.

[14] L. Hochberg, M. Serruya, G. Friehs, J. Mukand, M. Saleh, A. Caplan, A. Branner, D. Chen, R. Penn, and J. Donoghue. Neuronal ensemble control of prosthetic devices by a human with tetraplegia. *Nature*, 442(7099):164–171, 2006.

[15] H. Johansen-Berg, M. Rushworth, M. Bogdanovic, U. Kischka, S. Wimalaratna, and P. Matthews. The role of ipsilateral premotor cortex in hand movement after stroke. *Proceedings of the National Academy of Sciences*, 99(22):14518, 2002.

[16] M. Just, V. Cherkassky, S. Aryal, and T. Mitchell. A neurosemantic theory of concrete noun representation based on the underlying brain codes. 2010.

[17] E. Leuthardt, Z. Freudenberg, D. Bundy, and J. Roland. Microscale recording from human motor cortex: implications for minimally invasive electrocorticographic brain-computer interfaces. *Journal of Neurosurgery: Pediatrics*, 27(1), 2009.

[18] K. Miller, S. Makeig, A. Hebb, R. Rao, M. Dennijs, and J. Ojemann. Cortical electrode localization from x-rays and simple mapping for electrocorticographic research: The. *Journal of neuroscience methods*, 162(1-2):303–308, 2007.

[19] D. Moran and A. Schwartz. Motor cortical representation of speed and direction during reaching. *Journal of Neurophysiology*, 82(5):2676, 1999.

[20] H. Nakayama, H. Jørgensen, H. Raaschou, and T. Olsen. Recovery of upper extremity function in stroke patients: the copenhagen stroke study. *Archives of physical medicine and rehabilitation*, 75(4):394, 1994.

[21] J. Newton, A. Sunderland, and P. Gowland. fmri signal decreases in ipsilateral primary motor cortex during unilateral hand movements are related to duration and side of movement. *Neuroimage*, 24(4):1080–1087, 2005.

[22] R. Nyberg-Hansen and A. Brodal. Sites of termination of corticospinal fibers in the cat. an experimental study with silver impregnation methods. *The Journal of Comparative Neurology*, 120(3):369–391, 2004.

[23] S. Petersen, P. Fox, M. Posner, M. Mintum, and M. Raichle. Positron emission tomographic studies of the cortical anatomy of single-word processing. *Cognitive psychology: key readings*, page 109, 2004.

[24] G. Pfurtscheller and A. Aranibar. Event-related cortical desynchronization detected by power measurements of scalp EEG* 1. *Electroencephalography and Clinical Neurophysiology*, 42(6):817–826, 1977.

[25] G. Pfurtscheller, C. Guger, G. Muller, G. Krausz, and C. Neuper. Brain oscillations control hand orthosis in a tetraplegic. *Neuroscience letters*, 292(3):211–214, 2000.

[26] S. Ryali and V. Menon. Feature selection and classification of fmri data using logistic regression with l1 norm regularization. *NeuroImage*, 47:S57, 2009.

[27] G. Schalk, D. McFarland, T. Hinterberger, N. Birbaumer, and J. Wolpaw. Bci2000: a general-purpose brain-computer interface system. *IEEE Transactions on Biomedical Engineering*, 51(6):1034–1043, 2004.

[28] R. Seitz, P. Hoflich, F. Binkofski, L. Tellmann, H. Herzog, and H. Freund. Role of the premotor cortex in recovery from middle cerebral artery infarction. *Archives of neurology*, 55(8):1081, 1998.

[29] H. Shibasaki and M. Kato. Movement-associated cortical potentials with unilateral and bilateral simultaneous hand movement. *Journal of Neurology*, 208(3):191–199, 1975.

[30] R. Srinivasan, P. Nunez, R. Silberstein, E. Inc, and O. Eugene. Spatial filtering and neocortical dynamics: estimates of eeg coherence. *IEEE Transactions on Biomedical Engineering*, 45(7):814–826, 1998.

[31] C. Tallon-Baudry. Oscillatory synchrony and human visual cognition. *Journal of Physiology-Paris*, 97(2-3):355–363, 2003.

[32] J. Tanji, K. Okano, and K. Sato. Neuronal activity in cortical motor areas related to ipsilateral, contralateral, and bilateral digit movements of the monkey. *Journal of neurophysiology*, 60(1):325, 1988.

[33] I. Tarkka and M. Hallett. Cortical topography of premotor and motor potentials preceding self-paced, voluntary movement of dominant and non-dominant hands. *Electroencephalography and Clinical Neurophysiology*, 75(1-2):36–43, 1990.

[34] D. Taylor and A. Schwartz. Direct cortical control of 3d neuroprosthetic devices. Aug. 17 2004. US Patent App. 10/495,207.

[35] A. Turton, S. Wroe, N. Trepte, C. Fraser, and R. Lemon. Contralateral and ipsilateral emg responses to transcranial magnetic stimulation during recovery of arm and hand function after stroke. *Electroencephalography and Clinical Neurophysiology/Electromyography and Motor Control*, 101(4):316–328, 1996.

[36] T. Verstynen, J. Diedrichsen, N. Albert, P. Aparicio, and R. Ivry. Ipsilateral motor cortex activity during unimanual hand movements relates to task complexity. *Journal of Neurophysiology*, 93(3):1209, 2005.

[37] C. Weiller, F. Chollet, K. Friston, R. Wise, and R. Frackowiak. Functional reorganization of the brain in recovery from striatocapsular infarction in man. *Annals of Neurology*, 31(5):463–472, 2004.

[38] K. Wisneski, N. Anderson, G. Schalk, M. Smyth, D. Moran, and E. Leuthardt. Unique cortical physiology associated with ipsilateral hand movements and neuroprosthetic implications. *Stroke*, 39(12):3351, 2008.

[39] J. Wolpaw, N. Birbaumer, D. McFarland, G. Pfurtscheller, and T. Vaughan. Brain-computer interfaces for communication and control. *Clinical neurophysiology*, 113(6):767–791, 2002.

[40] J. Wolpaw and D. McFarland. Control of a two-dimensional movement signal by a noninvasive brain-computer interface in humans. *Proceedings of the National Academy of Sciences of the United States of America*, 101(51):17849, 2004.

